# Channel Noise in Excitable Neuronal Membranes

**Amit Manwani,* Peter N. Steinmetz and Christof Koch**
Computation and Neural Systems Program, M-S 139-74
California Institute of Technology Pasadena, CA 91125
{*quixote,peter,koch*} *@klab.caltech.edu*

## Abstract

Stochastic fluctuations of voltage-gated ion channels generate current and voltage noise in neuronal membranes. This noise may be a critical determinant of the efficacy of information processing within neural systems. Using Monte-Carlo simulations, we carry out a systematic investigation of the relationship between channel kinetics and the resulting membrane voltage noise using a stochastic Markov version of the Mainen-Sejnowski model of dendritic excitability in cortical neurons. Our simulations show that kinetic parameters which lead to an increase in membrane excitability (increasing channel densities, decreasing temperature) also lead to an increase in the magnitude of the sub-threshold voltage noise. Noise also increases as the membrane is depolarized from rest towards threshold. This suggests that channel fluctuations may interfere with a neuron's ability to function as an integrator of its synaptic inputs and may limit the reliability and precision of neural information processing.

## 1 Introduction

Voltage-gated ion channels undergo random transitions between different conformational states due to thermal agitation. Generally, these states differ in their ionic permeabilities and the stochastic transitions between them give rise to conductance fluctuations which are a source of membrane noise [1]. In excitable cells, voltage-gated channel noise can contribute to the generation of spontaneous action potentials [2, 3], and the variability of spike timing [4]. Channel fluctuations can also give rise to bursting and chaotic spiking dynamics in neurons [5, 6].

Our interest in studying membrane noise is based on the thesis that noise ultimately limits the ability of neurons to transmit and process information. To study this problem, we combine methods from information theory, membrane biophysics and compartmental neuronal modeling to evaluate ability of different biophysical components of a neuron, such as the synapse, the dendritic tree, the soma and so on, to transmit information [7, 8, 9]. These neuronal components differ in the type, density, and kinetic properties of their constituent ion channels. Thus, measuring the impact of these differences on membrane noise rep-

resents a fundamental step in our overall program of evaluating information transmission within and between neurons.

Although information in the nervous system is mostly communicated in the form of action potentials, we first direct our attention to the study of sub-threshold voltage fluctuations for three reasons. Firstly, voltage fluctuations near threshold can cause variability in spike timing and thus directly influence the reliability and precision of neuronal activity. Secondly, many computations putatively performed in the dendritic tree (coincidence detection, multiplication, synaptic integration and so on) occur in the sub-threshold regime and thus are likely to be influenced by sub-threshold voltage noise. Lastly, several sensory neurons in vertebrates and invertebrates are non-spiking and an analysis of voltage fluctuations can be used to study information processing in these systems as well.

Extensive investigations of channel noise were carried out prior to the advent of the patch-clamp technique in order to provide indirect evidence for the existence of single ion channels (see [1] for an excellent review). More recently, theoretical studies have focused on the effect of random channel fluctuations on spike timing and reliability of individual neurons [4], as well as their effect on the dynamics of interconnected networks of neurons [5, 6]. In this paper, we determine the effect of varying the kinetic parameters, such as channel density and the rate of channel transitions, on the magnitude of sub-threshold voltage noise in an iso-potential membrane patches containing stochastic voltage-gated ion channels using Monte-Carlo simulations. The simulations are based on the Mainen-Sejnowski (MS) kinetic model of active channels in the dendrites of cortical pyramidal neurons [10]. By varying two model parameters (channel densities and temperature), we investigate the relationship between excitability and noise in neuronal membranes. By linearizing the channel kinetics, we derive analytical expressions which provide closed-form estimates of noise magnitudes; we contrast the results of the simulations with the linearized expressions to determine the parameter range over which they can be used.

## 2   Monte-Carlo Simulations

Consider an iso-potential membrane patch containing voltage-gated $K^+$ and $Na^+$ channels and leak channels,

$$-C \frac{dV_m}{dt} = g_K (V_m - E_K) + g_{Na} (V_m - E_{Na}) + g_L (V_m - E_L) + I_{inj} \qquad (1)$$

where $C$ is the membrane capacitance and $g_K$ ($g_{Na}$, $g_L$) and $E_K$ ($E_{Na}$, $E_L$) denote the $K^+$($Na^+$, leak) conductance and the $K^+$($Na^+$, leak) reversal potential respectively. Current injected into the patch is denoted by $I_{inj}$, with the convention that inward current is negative. The channels which give rise to potassium and sodium conductances switch randomly between different conformational states with voltage-dependent transition rates. Thus, $g_K$ and $g_{Na}$ are voltage-dependent random processes and eq. 1 is a non-linear stochastic differential equation. Generally, ion channel transitions are assumed to be Markovian [11] and the stochastic dynamics of eq. 1 can be studied using Monte-Carlo simulations of finite-state Markov models of channel kinetics.

Earlier studies have carried out simulations of stochastic versions of the classical Hodgkin-Huxley kinetic model [12] to study the effect of conductance fluctuations on neuronal spiking [13, 2, 4]. Since we are interested in sub-threshold voltage noise, we consider a stochastic Markov version of a less excitable kinetic model used to describe dendrites of cortical neurons [10]. We shall refer to it as the Mainen-Sejnowski (MS) kinetic scheme. The $K^+$conductance is modeled by a single activation sub-unit (denoted by $n$) whereas the $Na^+$conductance is comprised of three identical activation sub-units (denoted by $m$) and one inactivation sub-unit (denoted by $h$). Thus, the stochastic discrete-state Markov models of the $K^+$and $Na^+$channel have 2 and 8 states respectively (shown in Fig. 1). The

single channel conductances and the densities of the ion channels ($K^+$,$Na^+$) are denoted by ($\gamma_K$,$\gamma_{Na}$) and ($\eta_K$,$\eta_{Na}$) respectively. Thus, $g_K$ and $g_{Na}$) are given by the products of the respective single channel conductances and the corresponding numbers of channels in the conducting states.

**A**                    **B**

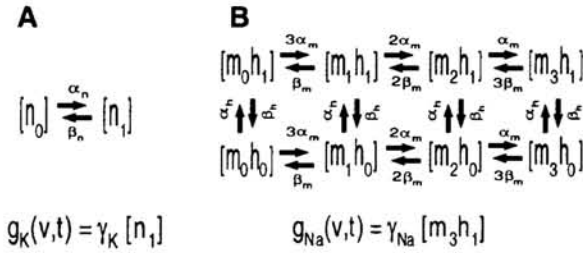

Figure 1: Kinetic scheme for the voltage-gated Mainen-Sejnowski $K^+$(**A**) and $Na^+$(**B**) channels. $n_0$ and $n_1$ represent the closed and open states of $K^+$channel. $m_{0-2}h_1$ represent the 3 closed states, $m_{0-3}h_0$ the four inactivated states and $m_3h_1$ the open state of the $Na^+$ channel.

We performed Monte-Carlo simulations of the MS kinetic scheme using a fixed time step of $\Delta t = 10$ $\mu$sec. During each step, the number of sub-units undergoing transitions between states $i$ and $j$ was determined by drawing a pseudo-random binomial deviate (bnldev subroutine [14] driven by the ran2 subroutine of the $2^{nd}$ edition) with $N$ equal to the number of sub-units in state $i$ and $p$ given by the conditional probability of the transition between $i$ and $j$. After updating the number of channels in each state, eq. 1 was integrated using fourth order Runge-Kutta integration with adaptive step size control [14]. During each step, the channel conductances

were held at the fixed value corresponding to the new numbers of open channels. (See [4] for details of this procedure).

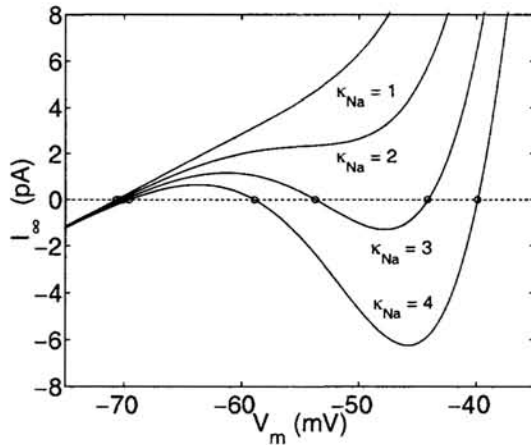

Figure 2: Steady-state I-V curves for different multiples ($\kappa_{Na}$) of the nominal MS $Na^+$channel density. Circles indicate locations of fixed-points in the absence of current injection.

Due to random channel transitions, the membrane voltage fluctuates around the steady-state resting membrane voltage $V_{rest}$. By varying the magnitude of the constant injected current $I_{inj}$, the steady-state voltage can be varied over a broad range, which depends on the channel densities. The current required to maintain the membrane at a holding voltage $V_{hold}$ can be determined from the steady-state I-V curve of the system, as shown in Fig. 2. Voltages for which the slope of the I-V curve is negative cannot be maintained as steady-states. By injecting an external current to offset the total membrane current, a fixed point in the negative slope region can be obtained but since the fixed point is unstable, any perturbation, such as a stochastic ion channel opening or closing, causes the system to be driven to the closest sta-

ble fixed point. We measured sub-threshold voltage noise only for stable steady-state holding voltages. A typical voltage trace from our simulations is shown in Fig. 3. To estimate the standard deviation of the voltage noise accurately, simulations were performed for a total of 492 seconds, divided into 60 blocks of 8.2 seconds each, for each steady-state value.

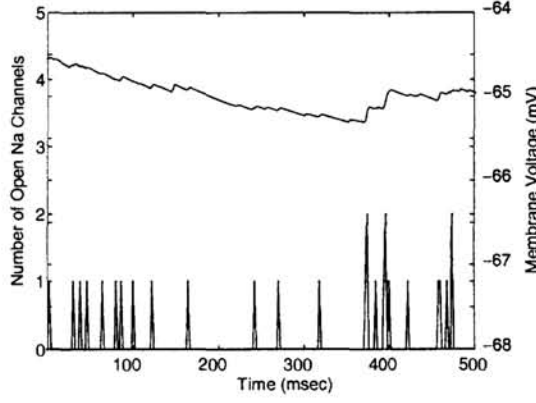

Figure 3: Monte-Carlo simulations of a 1000 $\mu m^2$ membrane patch with stochastic $Na^+$ and deterministic $K^+$ channels with MS kinetics. Bottom record shows the number of open $Na^+$ channels as a function of time. Top trace shows the corresponding fluctuations of the membrane voltage. Summary of nominal MS parameters: $C_m = 0.75$ $\mu F/cm^2$, $\eta_K = 1.5$ channels/$\mu m^2$, $\eta_{Na} = 2$ channels/$\mu m^2$, $E_K = -90$ mV, $E_{Na} = 60$ mV, $E_L = -70$ mV, $g_L = 0.25$ pS/$\mu m^2$, $\gamma_K = \gamma_{Na} = 20$ pS.

## 3   Linearized Analysis

The non-linear stochastic differential equation (eq. 1) cannot be solved analytically. However, one can linearize it by expressing the ionic conductances and the membrane voltage as small perturbations ($\delta$) around their steady-state values:

$$-C\frac{d\delta V_m}{dt} = (g_K^o + g_{Na}^o + g_L)\delta V_m + (V_m^o - E_K)\delta g_K + (V_m^o - E_{Na})\delta g_{Na} \quad (2)$$

where $g_K^o$ and $g_{Na}^o$ denote the values of the ionic conductances at the steady-state voltage $V^o$. $G = g_K^o + g_{Na}^o + g_L$ is the total steady-state patch conductance. Since the leak channel conductance is constant, $\delta g_L = 0$. On the other hand, $\delta g_K$ and $\delta g_{Na}$ depend on $\delta V$ and $t$. It is known that, to first order, the voltage- and time-dependence of active ion channels can be modeled as phenomenological impedances [15, 16]. Fig. 4 shows the linearized equivalent circuit of a membrane patch, given by the parallel combination of the capacitance $C$, the conductance $G$ and three series RL branches representing phenomenological models of $K^+$ activation, $Na^+$ activation and $Na^+$ inactivation.

$$I_n = \tilde{g}_K(E_K - V_m^o) + \tilde{g}_{Na}(E_{Na} - V_m^o) \quad (3)$$

represents the current noise due to fluctuations in the channel conductances (denoted by $\tilde{g}_K$ and $\tilde{g}_{Na}$) at the membrane voltage $V_m^o$ (also referred to as holding voltage $V_{hold}$). The details of the linearization are provided [16]. The complex admittance (inverse of the impedance) of Fig. 4 is given by,

$$Y(f) = G + j2\pi fC + \frac{1}{r_n + j2\pi fl_n} + \frac{1}{r_m + j2\pi fl_m} + \frac{1}{r_h + j2\pi fl_h} \quad (4)$$

The variance of the voltage fluctuations $\sigma_V^2$ can be computed as,

$$\sigma_V^2 = \int_{-\infty}^{\infty} df \frac{S_{In}(f)}{|Y(f)|^2} \quad (5)$$

where the power spectral density of $I_n$ is given by the sum of the individual channel current noise spectra, $S_{In}(f) = S_{IK}(f) + S_{INa}(f)$.

For the MS scheme, the autocovariance of the $K^+$ current noise for patch of area $A$, clamped at a voltage $V_m^o$, can be derived using [1, 11],

$$C_{IK}(t) = A\eta_K\gamma_K^2(V_m^o - E_K)^2 n_\infty(1 - n_\infty)e^{-|t|/\tau_n} \quad (6)$$

where $n_\infty$ and $\tau_n$ respectively denote the steady-state probability and time constant of the $K^+$ activation sub-unit at $V_m^o$. The power spectral density of the $K^+$ current noise $S_{IK}(f)$ can be obtained from the Fourier transform of $C_{IK}(t)$,

$$S_{IK}(f) = \frac{2A\eta_K\gamma_K^2(V_m^o - E_K)^2 n_\infty\tau_n}{1 + (2\pi f\tau_n)^2} \quad (7)$$

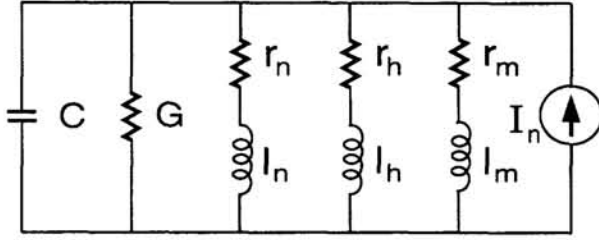

Figure 4: Linearized circuit of the membrane patch containing stochastic voltage-gated ion channels. $C$ denotes the membrane capacitance, $G$ is the sum of the steady-state conductances of the channels and the leak. $r_i$'s and $l_i$'s denote the phenomenological resistances and inductances due to the voltage- and time-dependent ionic conductances.

Thus, $S_{IK}(f)$ is a single Lorentzian spectrum with cut-off frequency determined by $\tau_n$. Similarly, the auto-covariance of the MS Na$^+$ current noise can be written as [1],

$$C_{INa}(t) = A\,\eta_{Na}\,\gamma_{Na}^2\,(V_m^o - E_{Na})^2\,m_\infty^3\,h_\infty\left[m^3(t)\,h(t) - m_\infty^3\,h_\infty\right] \tag{8}$$

where

$$m(t) = m_\infty + (1 - m_\infty)\,e^{-t/\tau_m}, \quad h(t) = h_\infty + (1 - h_\infty)\,e^{-t/\tau_h} \tag{9}$$

As before, $m_\infty$ ($h_\infty$) and $\tau_m$ ($\tau_h$) are the open probability and the time constant of Na$^+$ activation (inactivation) sub-unit. The Na$^+$ current noise spectrum $S_{INa}(f)$ can be expressed as a sum of Lorentzian spectra with cut-off frequencies corresponding to the seven time constants $\tau_m, \tau_h, 2\,\tau_m, 3\,\tau_m, \tau_m + \tau_h, 2\,\tau_m + \tau_h$ and $3\,\tau_m + \tau_h$. The details of the derivations can be found in [8].

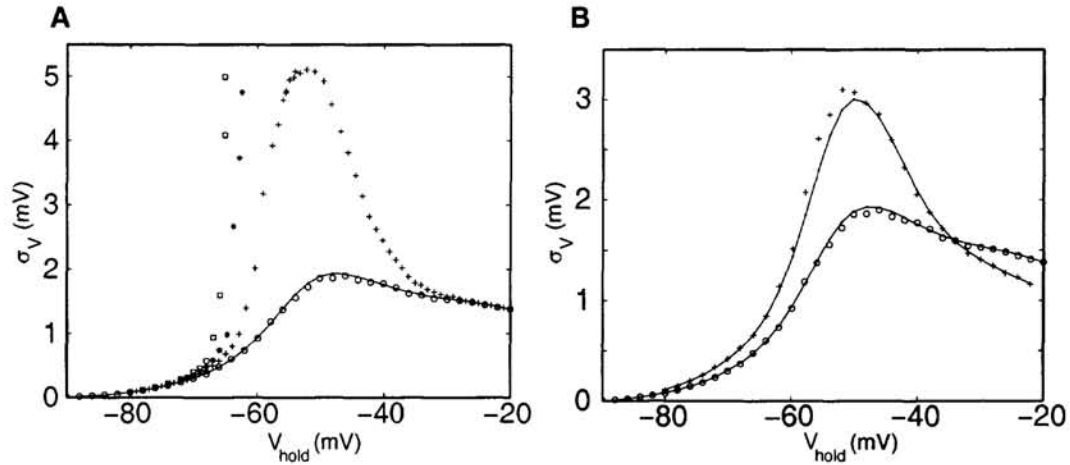

Figure 5: Standard deviation of the voltage noise $\sigma_V$ in a 1000 $\mu m^2$ patch as a function of the holding voltage $V_{hold}$. Circles denote results of the Monte-Carlo simulations for the nominal MS parameter values (see Fig. 3). The solid curve corresponds to the theoretical expression obtained by linearizing the channel kinetics. (**A**) Effect of increasing the sodium channel density by a factor (compared to the nominal value) of 2 (pluses), 3 (asterisks) and 4 (squares) on the magnitude of voltage noise. (**B**) Effect of increasing both the sodium and potassium channel densities by a factor of two (pluses).

## 4   Effect of Varying Channel Densities

Fig. 5 shows the voltage noise for a 1000 $\mu m^2$ patch as a function of the holding voltage for different values of the channel densities. Noise increases as the membrane is depolarized from rest towards -50 mV and the rate of increase is higher for higher Na$^+$ densities. The range of $V_{hold}$ for sub-threshold behavior extends up to -20 mV for nominal densities,

but does not exceed -60 mV for higher $Na^+$ densities. For moderate levels of depolariza-
tion, an increase in the magnitude of the ionic current noise with voltage is the dominant
factor which leads to an increase in voltage noise; for higher voltages phenomenological
impedances are large and shunt away the current noise. Increasing $Na^+$ density increases
voltage noise, whereas, increasing $K^+$ density causes a decrease in noise magnitude (com-
pare Fig. 5A and 5B). We linearized closed-form expressions provide accurate estimates
of the noise magnitudes when the noise is small (of the order 3 mV).

## 5   Effect of Varying Temperature

Fig. 6 shows that voltage noise decreases with
temperature. To model the effect of temperature,
transition rates were scaled by a factor $Q_{10}^{\Delta T/10}$
($Q_{10} = 2.3$ for $K^+$, $Q_{10} = 3$ for $Na^+$). Tem-
perature increases the rates of channel transitions
and thus the bandwidth of the ionic current noise
fluctuations. The magnitude of the current noise,
on the other hand, is independent of temperature.
Since the membrane acts as a low-pass RC fil-
ter (at moderately depolarized voltages, the phe-
nomenological inductances are small), increasing
the bandwidth of the noise results in lower volt-
age noise as the high frequency components are
filtered out.

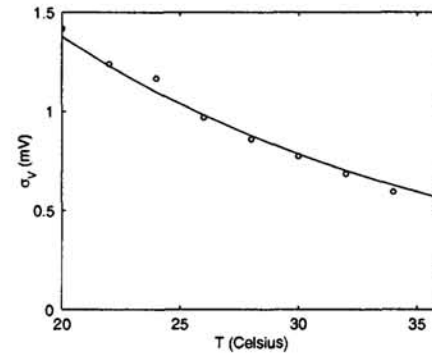

Figure 6: $\sigma_V$ as a function of tem-
perature for a 1000 $\mu m^2$ patch with
MS kinetics ($V_{hold} = -60$ mV). Circles
denote Monte-Carlo simulations, solid
curve denotes linearized approximation.

## 6   Conclusions

We studied sub-threshold voltage noise due to stochastic ion channel fluctuations in an iso-
potential membrane patch with Mainen-Sejnowski kinetics. For the MS kinetic scheme,
noise increases as the membrane is depolarized from rest, up to the point where the phe-
nomenological impedances due to the voltage- and time-dependence of the ion channels
become large and shunt away the noise. Increasing $Na^+$ channel density increases both the
magnitude of the noise and its rate of increase with membrane voltage. On the other hand,
increasing the rates of channel transitions by increasing temperature, leads to a decrease
in noise. It has previously been shown that neural excitability increases with $Na^+$ channel
density [17] and decreases with temperature [15]. Thus, our findings suggest that an in-
crease in membrane excitability is inevitably accompanied by an increase in the magnitude
of sub-threshold voltage noise fluctuations. The magnitude and the rapid increase of volt-
age noise with depolarization suggests that channel fluctuations can contribute significantly
to the variability in spike timing [4] and the stochastic nature of ion channels may have a
significant impact on information processing within individual neurons. It also potentially
argues against the conventional role of a neuron as integrator of synaptic inputs [18], as
the the slow depolarization associated with integration of small synaptic inputs would be
accompanied by noise, making the membrane voltage a very unreliable indicator of the
integrated inputs. We are actively investigating this issue more carefully.

When the magnitudes of the noise and the phenomenological impedances are small, the
non-linear kinetic schemes are well-modeled by their linearized approximations. We have
found this to be valid for other kinetic schemes as well [19]. These analytical approxi-
mations can be used to study noise in more sophisticated neuronal models incorporating
realistic dendritic geometries, where Monte-Carlo simulations may be too computationally
intensive to use.

## Acknowledgments

This work was funded by NSF, NIMH and the Sloan Center for Theoretical Neuroscience. We thank our collaborators Michael London, Idan Segev and Yosef Yarom for their invaluable suggestions.

## Footnotes

*http://www.klab.caltech.edu/~quixote

## References

[1] DeFelice L.J. (1981). *Introduction to Membrane Noise.* Plenum Press: New York, New York.

[2] Strassberg A.F. & DeFelice L.J. (1993). Limitations of the Hodgkin-Huxley formalism: effect of single channel kinetics on transmembrane voltage dynamics. *Neural Computation*, 5:843–855.

[3] Chow C. & White J. (1996). Spontaneous action potentials due to channel fluctuations. *Biophy. J.*, 71:3013–3021.

[4] Schneidman E., Freedman B. & Segev I. (1998). Ion-channel stochasticity may be critical in determining the reliability and precision of spike timing. *Neural Computation*, 10:1679–1703.

[5] DeFelice L.J. & Isaac A. (1992). Chaotic states in a random world. *J. Stat. Phys.*, 70:339–352.

[6] White J.A., Budde T. & Kay A.R. (1995). A bifurcation analysis of neuronal subthreshold oscillations. *Biophy. J.*, 69:1203–1217.

[7] Manwani A. & Koch C. (1998). Synaptic transmission: An information-theoretic perspective. In: Jordan M., Kearns M.S. & Solla S.A., eds., *Advances in Neural Information Processing Systems 10.* pp 201-207. MIT Press: Cambridge, Massachusetts.

[8] Manwani A. & Koch C. (1999). Detecting and estimating signals in noisy cable structures: I. Neuronal noise sources. *Neural Computation*. In press.

[9] Manwani A. & Koch C. (1999). Detecting and estimating signals in noisy cable structures: II. Information-theoretic analysis. *Neural Computation*. In press.

[10] Mainen Z.F. & Sejnowski T.J. (1995). Reliability of spike timing in neocortical neurons. *Science*, 268:1503–1506.

[11] Johnston D. & Wu S.M. (1995). *Foundations of Cellular Neurophysiology.* MIT Press: Cambridge, Massachusetts.

[12] Hodgkin A.L. & Huxley A.F. (1952). A quantitative description of membrane current and its application to conduction and excitation in nerve. *J. Physiol. (London)*, 117:500–544.

[13] Skaugen E. & Walløe L. (1979). Firing behavior in a stochastic nerve membrane model based upon the Hodgkin-Huxley equations. *Acta Physiol. Scand.*, 107:343–363.

[14] Press W.H., Teukolsky S.A., Vetterling W.T. & Flannery B.P. (1992). *Numerical Recipes in C: The Art of Scientific Computing.* Cambridge University Press, second edn.

[15] Mauro A., Conti F., Dodge F. & Schor R. (1970). Subthreshold behavior and phenomenological impedance of the squid giant axon. *J. Gen. Physiol.*, 55:497–523.

[16] Koch C. (1984). Cable theory in neurons with active, linearized membranes. *Biol. Cybern.*, 50:15–33.

[17] Sabah N.H. & Leibovic K.N. (1972). The effect of membrane parameters on the properties of the nerve impulse. *Biophys. J.*, 12:1132–44.

[18] Shadlen M.N. & Newsome W.T. (1998). The variable discharge of cortical neurons: implications for connectivity, computation, and information coding. *J. Neurosci.*, 18:3870–3896.

[19] P. N. Steinmetz A. Manwani M.L. & Koch C. (1999). Sub-threshold voltage noise due to channel fluctuations in active neuronal membranes. In preparation.